# Support Vector Novelty Detection Applied to Jet Engine Vibration Spectra

**Paul Hayton**
Department of Engineering Science
University of Oxford, UK
*pmh@robots.ox.ac.uk*

**Bernhard Schölkopf**
Microsoft Research
1 Guildhall Street, Cambridge, UK
*bsc@scientist.com*

**Lionel Tarassenko**
Department of Engineering Science
University of Oxford, UK
*lionel@robots.ox.ac.uk*

**Paul Anuzis**
Rolls-Royce Civil Aero-Engines
Derby, UK

## Abstract

A system has been developed to extract diagnostic information from jet engine carcass vibration data. Support Vector Machines applied to novelty detection provide a measure of how unusual the shape of a vibration signature is, by learning a representation of normality. We describe a novel method for Support Vector Machines of including information from a second class for novelty detection and give results from the application to Jet Engine vibration analysis.

## 1 Introduction

Jet engines have a number of rigorous pass-off tests before they can be delivered to the customer. The main test is a vibration test over the full range of operating speeds. Vibration gauges are attached to the casing of the engine and the speed of each shaft is measured using a tachometer. The engine on the test bed is slowly accelerated from idle to full speed and then gradually decelerated back to idle. As the engine accelerates, the rotation frequency of the two (or three) shafts increases and so does the frequency of the vibrations caused by the shafts. A *tracked order* is the amplitude of the vibration signal in a narrow frequency band centered on a harmonic of the rotation frequency of a shaft, measured as a function of engine speed. It tracks the frequency response of the engine to the energy injected by the rotating shaft. Although there are usually some harmonics present, most of the energy in the vibration spectrum is concentrated in the fundamental tracked orders. These therefore constitute the "vibration signature" of the jet engine under test. It is very important to detect departures from the normal or expected shapes of these tracked orders as this provides very useful diagnostic information (for example, for the identification of out-of-balance conditions).

The detection of such abnormalities is ideally suited to the novelty detection paradigm for several reasons. Usually, there are far fewer examples of abnormal shapes than normal ones and often there may only be a single example of a particular type of abnormality in

the available database. More importantly, the engine under test may show up a type of abnormality *which has never been seen before* but which should not be missed. This is especially important in our current work where we are adapting the techniques developed for pass-off tests to in-flight monitoring.

With novelty detection, we first of all learn a description of normal vibration shapes by including only examples of normal tracked orders in the training data. Abnormal shapes in test engines are subsequently identified by testing for novelty against the description of normality.

In our previous work [2], we investigated the vibration spectra of a two-shaft jet engine, the Rolls-Royce Pegasus. In the available database, there were vibration spectra recorded from 52 normal engines (the training data) and from 33 engines with one or more unusual vibration feature (the test data). The shape of the tracked orders was encoded as a low-dimensional vector by calculating a weighted average of the vibration amplitude over six different speed ranges (giving an 18-D vector for three tracked orders). With so few engines available, the $K$-means clustering algorithm (with $K = 4$) was used to construct a very simple model of normality, following component-wise normalisation of the 18-D vectors.

The novelty of the vibration signature for a test engine was assessed as the shortest distance to one of the kernel centres in the clustering model of normality (each distance being normalised by the width associated with that kernel). When cumulative distributions of novelty scores were plotted both for normal (training) engines and test engines, there was little overlap found between the two distributions [2]. A significant shortcoming of the method, however, is the inability to rank engines according to novelty, since the shortest normalised distance is evaluated with respect to different cluster centres for different engines. In this paper, we re-visit the problem but for a new engine, the RB211-535. We argue that the SVM paradigm is ideal for novelty detection, as it provides an elegant distribution of normality, a direct indication of the patterns on the boundary of normality (the support vectors) and, perhaps most importantly, a ranking of "abnormality" according to distance to the separating hyperplane in feature space.

## 2 Support Vector Machines for Novelty Detection

Suppose we are given a set of "normal" data points $\mathbf{X} = \{\mathbf{x}_1, \ldots, \mathbf{x}_\ell\}$. In most novelty detection problems, this is all we have; however, in the following we shall develop an algorithm that is slightly more general in that it *can* also take into account some examples of abnormality, $\mathbf{Z} = \{\mathbf{z}_1, \ldots, \mathbf{z}_t\}$. Our goal is to construct a real-valued function which, given a previously unseen test point $\mathbf{x}$, charaterizes the "$\mathbf{X}$-ness" of the point $\mathbf{x}$, i.e. which takes large values for points similar to those in $\mathbf{X}$. The algorithm that we shall present below will return such a function, along with a threshold value, such that a prespecified fraction of $\mathbf{X}$ will lead to function values above threshold. In this sense we are estimating a region which captures a certain probability mass.

The present approach employs two ideas from support vector machines [6] which are crucial for their fine generalization performance even in high-dimensional tasks: maximizing a margin, and nonlinearly mapping the data into some *feature space F* endowed with a dot product. The latter need not be the case for the input domain $\mathcal{X}$ which may be a general set. The connection between the input domain and the feature space is established by a feature map $\Phi : \mathcal{X} \to F$, i.e. a map such that some simple kernel [1, 6]

$$k(\mathbf{x}, \mathbf{y}) = (\Phi(\mathbf{x}) \cdot \Phi(\mathbf{y})), \tag{1}$$

such as the Gaussian

$$k(\mathbf{x}, \mathbf{y}) = e^{-\|\mathbf{x}-\mathbf{y}\|^2/c}, \tag{2}$$

provides a dot product in the image of $\Phi$. In practice, we need not necessarily worry about $\Phi$, as long as a given $k$ satisfies certain positivity conditions [6].

As $F$ is a dot product space, we can use tools of linear algebra and geometry to construct algorithms in $F$, even if the input domain $\mathcal{X}$ is discrete. Below, we derive our results in $F$, using the following shorthands:

$$x_i = \Phi(\mathbf{x}_i), \; z_n = \Phi(\mathbf{z}_n) \tag{3}$$

$$X = \{x_1, \ldots, x_\ell\}, \; Z = \{z_1, \ldots, z_t\} \tag{4}$$

Indices $i$ and $j$ are understood to range over $1, \ldots, \ell$ (in compact notation: $i, j \in [\ell]$), similarly, $n, p \in [t]$. Bold face greek letters denote $\ell$-dimensional vectors whose components are labelled using normal face typeset.

In analogy to an algorithm recently proposed for the estimation of a distribution's support [5], we seek to separate $X$ from the centroid of $Z$ with a large margin hyperplane committing few training errors. Projections on the normal vector of the hyperplane then characterize the "$X$-ness" of test points, and the area where the decision function takes the value 1 can serve as an approximation of the support of $X$. While $X$ is the set of normal examples, the (possibly empty) set $Z$ thus only plays the role of, in some weak and possibly imprecise sense, modeling what the unknown "other" examples might look like.

The decision function is found by minimizing a weighted sum of a support vector type regularizer and an empirical error term depending on an overall margin variable $\rho$ and individual errors $\xi_i$,

$$\min_{w \in F, \xi \in \mathbb{R}^\ell, \rho \in \mathbb{R}} \quad \frac{1}{2} \|w\|^2 + \frac{1}{\nu\ell} \sum_i \xi_i - \rho \tag{5}$$

$$\text{subject to} \quad (w \cdot (x_i - \frac{1}{t} \sum_n z_n)) \geq \rho - \xi_i, \;\; \xi_i \geq 0. \tag{6}$$

The precise meaning of the parameter $\nu$ governing the trade-off between the regularizer and the training error will become clear later. Since nonzero slack variables $\xi_i$ are penalized in the objective function, we can expect that if $w$ and $\rho$ solve this problem, then the decision function

$$f(x) = \text{sgn}((w \cdot (x - \frac{1}{t} \sum_n z_n)) - \rho) \tag{7}$$

will be positive for many examples $x_i$ contained in $X$, while the SV type regularization term $\|w\|$ will still be small. This can be shown to correspond to a large margin of separation from $\frac{1}{t} \sum_n z_n$.

We next compute a dual form of this optimization problem. The details of the calculation, which uses standard techniques of constrained optimization, can be found in [4]. We introduce a Lagrangian and set the derivatives with respect to $w$ equal to zero, yielding in particular

$$w = \sum_i \alpha_i (x_i - \frac{1}{t} \sum_n z_n). \tag{8}$$

All patterns $\{\mathbf{x}_i : i \in [\ell], \; \alpha_i > 0\}$ are called Support Vectors. The expansion (8) turns the decision function (7) into a form which only depends on dot prducts, $f(x) = \text{sgn}((\sum_i \alpha_i(x_i - \frac{1}{t} \sum_n z_n) \cdot (x - \frac{1}{t} \sum_n z_n)) - \rho)$. By multiplying out the dot products, we obtain a form that can be written as a nonlinear decision function on the input domain $\mathcal{X}$ in terms of a kernel (1) (cf. (3)). A short calculation yields $f(\mathbf{x}) = \text{sgn}\left(\sum_i \alpha_i k(\mathbf{x}_i, \mathbf{x}) - \frac{1}{t} \sum_n k(\mathbf{z}_n, \mathbf{x}) + \frac{1}{t^2} \sum_{np} k(\mathbf{z}_n, \mathbf{z}_p) - \frac{1}{t} \sum_{in} \alpha_i k(\mathbf{z}_n, \mathbf{x}_i) - \rho\right)$. In the argument of the sgn, only the first two terms depend on $\mathbf{x}$, therefore we may absorb the

next terms in the constant $\rho$, which we have not fixed yet. To compute $\rho$ in the final form of the decision function

$$f(\mathbf{x}) = \text{sgn}\left(\sum_i \alpha_i k(\mathbf{x}_i, \mathbf{x}) - \frac{1}{t}\sum_n k(\mathbf{z}_n, \mathbf{x}) - \rho\right), \qquad (9)$$

we employ the Karush-Kuhn-Tucker (KKT) conditions of the optimization problem [6, e.g.]. They state that for points $\mathbf{x}_i$ where $0 < \alpha_i < 1/(\nu\ell)$, the inequality constraints (6) become equalities (note that in general, $\alpha_i \in [0, 1/(\nu\ell)]$), and the argument of the sgn in the decision function should equal 0, i.e. the corresponding $\mathbf{x}_i$ sits exactly on the hyperplane of separation.

The KKT conditions also imply that only those points $\mathbf{x}_i$ can have a nonzero $\alpha_i$ for which the first inequality constraint in (6) is precisely met; therefore the support vectors $\mathbf{x}_i$ with $\alpha_i > 0$ will often form but a small subset of $\mathbf{X}$.

Substituting (8) (the derivative of the Lagrangian by $w$) and the corresponding conditions for $\xi$ and $\rho$ into the Lagrangian, we can eliminate the primal variables to get the dual problem. A short calculation shows that it consists of minimizing the quadratic form

$$W(\alpha) = \frac{1}{2}\sum_{ij} \alpha_i \alpha_j \left(k(\mathbf{x}_i, \mathbf{x}_j) + q - q_j - q_i\right), \qquad (10)$$

where $q = \frac{1}{t^2}\sum_{np} k(\mathbf{z}_n, \mathbf{z}_p)$ and $q_j = \frac{1}{t}\sum_n k(\mathbf{x}_j, \mathbf{z}_n)$, subject to the constraints

$$0 \le \alpha_i \le 1/(\nu\ell), \quad \sum_i \alpha_i = 1. \qquad (11)$$

This convex quadratic program can be solved with standard quadratic programming tools. Alternatively, one can employ the SMO algorithm described in [3], which was found to approximately scale quadratically with the training set size.

To illustrate the idea presented in this section, figure 1 shows a 2D example of separating the data from the mean of another data set in feature space.

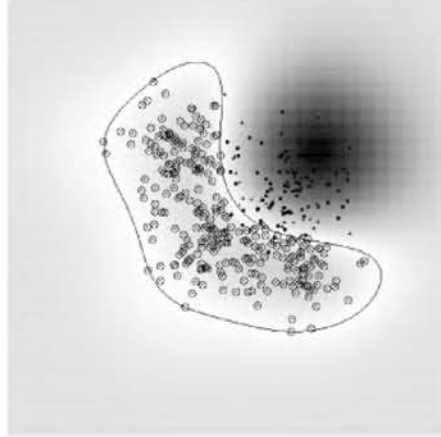

Figure 1: Separating one class of data from the mean of a second data set. The first class is a mixture of three gaussians; the SVM algorithm is used to find the hyperplane in feature space that separates the data from the second set (another Gaussian - the black dots). The image intensity represents the SVM output value which is the measure of novelty.

We next state a few theoretical results, beginning with a characterization of the influence of $\nu$. To this end, first note that the constraints (11) rule out solutions where $\nu > 1$, as in that case, the $\alpha_i$ cannot sum up to 1. Negative values of $\nu$ are ruled out, too, since they would amount to encouraging (rather than penalizing) training errors in (5). Therefore, in the primal problem (5) only $\nu \in (0, 1]$ makes sense. We shall now explain that $\nu$ actually characterizes how many points of $\mathbf{X}$ are allowed to lie outside the region where the decision function is positive. To this end, we introduce the term *outlier* to denote points $\mathbf{x}_i$ that have a nonzero slack variable $\xi_i$, i.e. points that lie outside of the estimated region. By the KKT conditions, all outliers are also support vectors; however there can be support vectors (sitting exactly on the margin) that are not outliers.

**Proposition 1 ($\nu$-property)** *Assume the solution of (5) satisfies $\rho \neq 0$. The following statements hold:*
*(i) $\nu$ is an upper bound on the fraction of outliers.*
*(ii) $\nu$ is a lower bound on the fraction of SVs.*
*(iii) Suppose the data (4) were generated independently from a distribution $P(x)$ which does not contain discrete components. Suppose, moreover, that the kernel is analytic and non-constant. With probability 1, asymptotically, $\nu$ equals both the fraction of SVs and the fraction of outliers.*

The proof can be found in [4]. We next state another desirable theoretical result:

**Proposition 2 (Resistance [3])** *Local movements of outliers parallel to $w$ do not change the hyperplane.*

Essentially, this result is due to the fact that the errors $\xi_i$ enter in the objective function only linearly. To determine the hyperplane, we need to find the (constrained) extremum of the objective function, and in finding the extremum, the derivatives are what counts. For the linear error term, however, those are constant, so they do not depend on how far away from the hyperplane an error point lies.

We conclude this section by noting that if $Z$ is empty, the algorithm is trying to separate the data from the origin in $F$, and both the decision function and the optimization problem reduce to what is described in [5].

## 3  Application of SVM to Jet Engine Pass-off Tests

The Support Vector machine algorithm for novelty detection is applied to the pass-off data from a set of 162 Rolls-Royce jet engines. The shape of the tracked order of interest is encoded by calculating a weighted average of the vibration amplitude over ten speed ranges, thereby generating a 10D shape vector. The available data was split into the following three sets:

- 99 Normal Engines to be used as training data;
- 40 Normal Engines to be used as validation data;
- 23 engines labelled as having at least one abnormal aspect in their vibration signature (the "test" data).

Using the training dataset, the SVM algorithm finds the hyperplane that separates the normal data from the origin in feature space with the largest margin. The number of support vectors gives an indication of how well the algorithm is generalising (if all data points were support vectors, the algorithm would have memorized the data). A Gaussian kernel was

used with a width $c = 40.0$ in equation 2 which was chosen by starting with a small kernel width (so that the algorithm memorizes the data), increasing the width and stopping when similar results are obtained on the training and validation data.

Cumulative novelty distributions are plotted for two different values of $\nu$ and these are shown in figure 2. The curves show a slight overlap between the normal and test engines. Although it is not given here, a ranking of the engines according to their novelty is also provided to the Rolls-Royce test engineers.

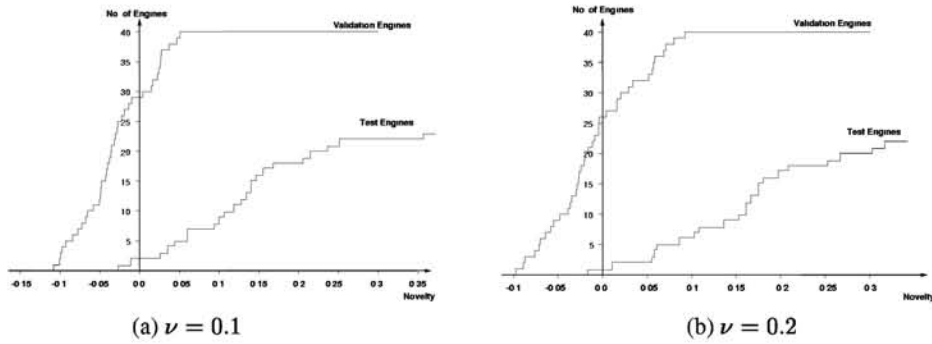

(a) $\nu = 0.1$                    (b) $\nu = 0.2$

Figure 2: Cumulative novelty distributions for two different values of $\nu$. The curves show that there is a slight overlap in the data; For $\nu = 0.1$, there are 11 validation engines over the SVM decision boundary and 2 test engines inside the boundary.

**Separating the Normal Engines from the Test Engines.**    In a retrospective analysis such as described in this paper (for which the test engines with unusual vibration signatures have already been identified as such by the Rolls-Royce experts), the SVM algorithm can be re-run to find the hyperplane that separates the normal data from the mean of the test data in feature space with the largest margin (instead of separating from the origin). The algorithm is trained on the 99 training engines and 22 of the 23 test engines. Each test engine is left out in turn and the algorithm re-trained to compute its novelty. Cumulative distributions are again plotted (see figure 3) and these show an improved separation between the two sets of engines. It should be noted however, that the improvement is less for the validation engines than for the training engines. Nevertheless, there is an improvement for the validation engines seen from the higher intersection of the distribution with the axis.

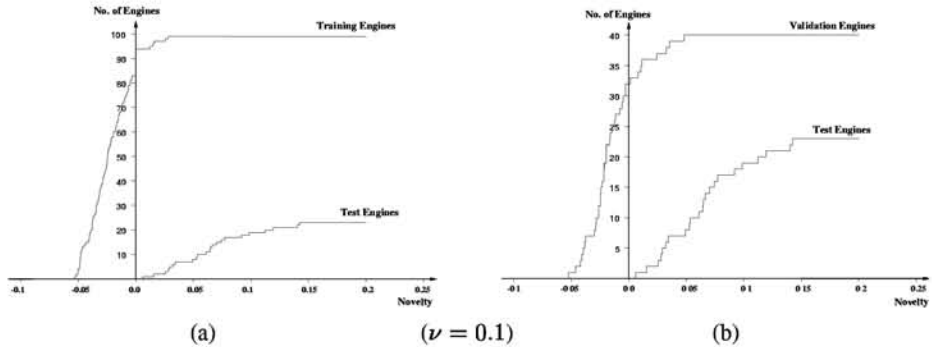

(a)              $(\nu = 0.1)$              (b)

Figure 3: Cumulative novelty distributions showing the variation of novelty with number of engines for (a) the training data versus the test data (each test engine omitted from the training phase in turn to compute its novelty) and (b) the validation data versus the test data.

# 4 Discussion

This paper has presented a novel application of Support Vector Machines and introduced a method for including information from a second data set when considering novelty detection. The results on the Jet Engine data show very good separation between normal and test engines. We believe Support Vector Machines are an ideal framework for novelty detection and indeed, we have obtained better results than with our previous clustering based algorithms for detecting novel Jet Engine signatures.

The present work builds on a previous algorithm for estimating a distribution's support [5]. That algorithm, separating the data from the origin in feature space, suffered from the drawback that the origin played a special role. One way to think of it is as a prior on where, in a novelty detection context, the unknown "other" class lies. The present work alleviates this problem by allowing for the possibility to separate from a point inferred from the data, either from the same class, or from some other data.

There is a concern that one could put forward about one of the variants of the presently proposed approach, namely about the case where $X$ and $Z$ are disjoint, and we are separating $X$ from $Z$'s centroid: why not actually train a full binary classifier separating $X$ from all examples from $Z$, rather that just from its mean? Indeed there might be situations where this is appropriate. More specifically, whenever $Z$ is *representative* of the instances of the other class that we expect to see in the future, then a binary classification is certainly preferable. However, there can be situations where $Z$ is not representative for the other class, for instance due to nonstationarity. $Z$ may even only consists of artificial examples. In this situation, the only *real* training examples are the positive ones. In this case, separating the data from the mean of some artificial, or non-representative examples, provides a way of taking into account *some* information from the other class which might work better than simply separating the positive data from the origin.

The philosophy behind our approach is the one advocated by [6]. If you are trying to solve a learning problem, do it directly, rather than solving a more general problem along the way. Applied to the estimation of a distribution's support, this means: do not first estimate a density and then threshold it to get an estimate of the support.

**Acknowledgments.** Thanks to John Platt, John Shawe-Taylor, Alex Smola and Bob Williamson for helpful discussions.

# References

[1] B. E. Boser, I. M. Guyon, and V. N. Vapnik. A training algorithm for optimal margin classifiers. In D. Haussler, editor, *Proceedings of the 5th Annual ACM Workshop on Computational Learning Theory*, pages 144–152, Pittsburgh, PA, July 1992. ACM Press.

[2] A. Nairac, N. Townsend, R. Carr, S. King, P. Cowley, and L. Tarassenko. A system for the analysis of jet engine vibration data. *Integrated Computer-Aided Engineering*, 6:53 – 65, 1999.

[3] B. Schölkopf, J. Platt, J. Shawe-Taylor, A.J. Smola, and R.C. Williamson. Estimating the support of a high-dimensional distribution. TR MSR 99 - 87, Microsoft Research, Redmond, WA, 1999.

[4] B. Schölkopf, J. Platt, and A.J. Smola. Kernel method for percentile feature extraction. TR MSR 2000 - 22, Microsoft Research, Redmond, WA, 2000.

[5] B. Schölkopf, R. C. Williamson, A. J. Smola, J. Shawe-Taylor, and J. C. Platt. Support vector method for novelty detection. In S.A. Solla, T.K. Leen, and K.-R. Müller, editors, *Advances in Neural Information Processing Systems 12*, pages 582–588. MIT Press, 2000.

[6] V. Vapnik. *The Nature of Statistical Learning Theory*. Springer, N.Y., 1995.
